# SPATIAL ORGANIZATION OF NEURAL NETWORKS: A PROBABILISTIC MODELING APPROACH

A. Stafylopatis
M. Dikaiakos
D. Kontoravdis
National Technical University of Athens, Department of Electrical Engineering, Computer Science Division, 157 73 Zographos, Athens, Greece.

## ABSTRACT

The aim of this paper is to explore the spatial organization of neural networks under Markovian assumptions, in what concerns the behaviour of individual cells and the interconnection mechanism. Space-organizational properties of neural nets are very relevant in image modeling and pattern analysis, where spatial computations on stochastic two-dimensional image fields are involved. As a first approach we develop a random neural network model, based upon simple probabilistic assumptions, whose organization is studied by means of discrete-event simulation. We then investigate the possibility of approximating the random network's behaviour by using an analytical approach originating from the theory of general product-form queueing networks. The neural network is described by an open network of nodes, in which customers moving from node to node represent stimulations and connections between nodes are expressed in terms of suitably selected routing probabilities. We obtain the solution of the model under different disciplines affecting the time spent by a stimulation at each node visited. Results concerning the distribution of excitation in the network as a function of network topology and external stimulation arrival pattern are compared with measures obtained from the simulation and validate the approach followed.

## INTRODUCTION

Neural net models have been studied for many years in an attempt to achieve brain-like performance in computing systems. These models are composed of a large number of interconnected computational elements and their structure reflects our present understanding of the organizing principles of biological nervous systems. In the begining, neural nets, or other equivalent models, were rather intended to represent the logic arising in certain situations than to provide an accurate description in a realistic context. However, in the last decade or so the knowledge of what goes on in the brain has increased tremendously. New discoveries in natural systems, make it now reasonable to examine the possibilities of using modern technology in order to synthesige systems that have some of the properties of real neural systems [8,9,10,11].

In the original neural net model developed in 1943 by McCulloch and Pitts [1,2] the network is made of many interacting components, known as the "McCulloch-Pitts cells" or "formal neurons", which are simple logical units with two possible states changing state accord-

ing to a threshold function of their inputs. Related automata models have been used later for gene control systems (genetic networks) [3], in which genes are represented as binary automata changing state according to boolean functions of their inputs. Boolean networks constitute a more general model, whose dynamical behaviour has been studied extensively. Due to the large number of elements, the exact structure of the connections and the functions of individual components are generally unknown and assumed to be distributed at random. Several studies on these random boolean networks [5,6] have shown that they exhibit a surprisingly stable behaviour in what concerns their temporal and spatial organization. However, very few formal analytical results are available, since most studies concern statistical descriptions and computer simulations.

The temporal and spatial organization of random boolean networks is of particular interest in the attempt of understanding the properties of such systems, and models originating from the theory of stochastic processes [13] seem to be very useful. Spatial properties of neural nets are most important in the field of image recognition [12]. In the biological eye, a level-normalization computation is performed by the layer of horizontal cells, which are fed by the immediately preceding layer of photoreceptors. The horizontal cells take the outputs of the receptors and average them spatially, this average being weighted on a nearest-neighbor basis. This procedure corresponds to a mechanism for determining the brightness level of pixels in an image field by using an array of processing elements. The principle of local computation is usually adopted in models used for representing and generating textured images. Among the stochastic models applied to analyzing the parameters of image fields, the random Markov field model [7,14] seems to give a suitably structured representation, which is mainly due to the application of the markovian property in space. This type of modeling constitutes a promising alternative in the study of spatial organization phenomena in neural nets.

The approach taken in this paper aims to investigate some aspects of spatial organization under simple stochastic assumptions. In the next section we develop a model for random neural networks assuming boolean operation of individual cells. The behaviour of this model, obtained through simulation experiments, is then approximated by using techniques from the theory of queueing networks. The approximation yields quite interesting results as illustrated by various examples.

## THE RANDOM NETWORK MODEL

We define a random neural network as a set of elements or cells, each one of which can be in one of two different states: firing or quiet. Cells are interconnected to form an NxN grid, where each grid point is occupied by a cell. We shall consider only connections between neighbors, so that each cell is connected to 4 among the other cells: two input and two output cells (the output of a cell is equal to its internal state and it is sent to its output cells which use it as one of their inputs). The network topology is thus specified

by its incidence matrix A of dimension MxM, where $M=N^2$. This matrix takes a simple form in the case of neighbor-connection considered here. We further assume a periodic structure of connections in what concerns inputs and outputs; we will be interested in the following two types of networks depending upon the period of reproduction for elementary square modules [5], as shown in Fig.1:
- Propagative nets (Period 1)
- Looping nets     (Period 2)

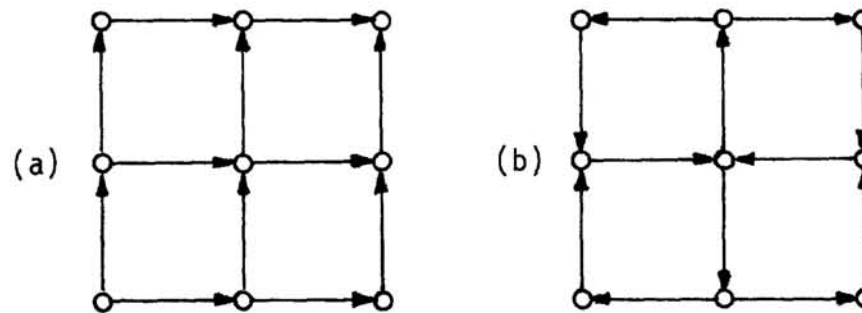

Fig.1. (a) Propagative connections, (b) Looping connections

At the edges of the grid, circular connections are established (modulo N), so that the network can be viewed as supported by a torus.
     The operation of the network is non-autonomous: changes of state are determined by both the interaction among cells and the influence of external stimulations. We assume that stimulations arrive from the outside world according to a Poisson process with parameter $\lambda$. Each arriving stimulation is associated with exactly one cell of the network; the cell concerned is determined by means of a given discrete probability distribution $q_i$ ($1 \leq i \leq M$), considering an one-dimensional labeling of the M cells.
     The operation of each individual cell is asynchronous and can be described in terms of the following rules:
- A quiet cell moves to the firing state if it receives an arriving stimulation or if a boolean function of its inputs becomes true.
- A firing cell moves to the quiet state if a boolean function of its inputs becomes false.
- Changes of state imply a reaction delay of the cell concerned; these delays are independent identically distributed random variables following a negative exponential distribution with parameter $\gamma$.
According to these rules, the operation of a cell can be viewed as illustrated by Fig.2, where the horizontal axis represents time and the numbers 0,1,2 and 3 represent phases of an operation cycle. Phases 1 and 3, as indicated in Fig.2, correspond to reaction delays. In this sense, the quiet and firing states, as defined in the begining of this section, represent the aggregates of phases 0,1 and 2,3 respectively. External stimulations affect the receiving cell only when it is in phase 0; otherwise we consider that the stimulation is lost. In the same way, we assume that changes of the value of the input boolean function do not affect the operation of the cell during phases 1 and 3. The conditions are checked only at the end of the respective reaction delay.

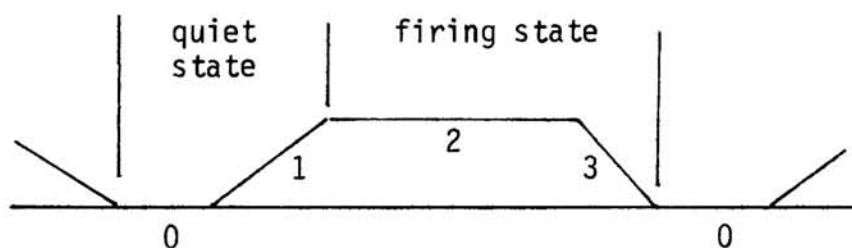

Fig.2. Changes of state for individual cells

The above defined model includes some features of the original McCulloch-Pitts cells [1,2]. In fact, it represents an asynchronous counterpart of the latter, in which boolean functions are considered instead of threshold functions. However, it can be shown that any McCulloch and Pitts' neural network can be implemented by a boolean network designed in an appropriate fashion [5]. In what follows, we will consider that the firing condition for each individual cell is determined by an "or" function of its inputs.

Under the assumptions adopted, the evolution of the network in time can be described by a continuous-parameter Markov process. However, the size of the state-space and the complexity of the system are such that no analytical solution is tractable. The spatial organization of the network could be expressed in terms of the steady-state probability distribution for the Markov process. A more useful representation is provided by the marginal probability distributions for all cells in the network, or equivalently by the probability of being in the firing state for each cell. This measure expresses the level of excitation for each point in the grid.

We have studied the behaviour of the above model by means of simulation experiments for various cases depending upon the network size, the connection type, the distribution of external stimulation arrivals on the grid and the parameters $\lambda$ and $\gamma$. Some examples are illustrated in the last section, in comparison with results obtained using the approach discussed in the next section. The estimations obtained concern the probability of being in the firing state for all cells in the network. The simulation was implemented according to the "batched means" method; each run was carried out until the width of the 95% confidence interval was less that 10% of the estimated mean value for each cell, or until a maximum number of events had been simulated depending upon the size of the network.

## THE ANALYTICAL APPROACH

The neural network model considered in the previous section exhibited the markovian property in both time and space. Markovianity in space, expressed by the principle of "neighbor-connections", is the basic feature of Markov random fields [7,14], as already discussed. Our idea is to attempt an approximation of the random neural network model by using a well-known model, which is markovian in time, and applying the constraint of markovianity in space. The model considered is an open queueing network, which belongs to the general class of queueing networks admitting a product-form solution [4]. In fact, one could distinguish several common features in the two network models.

A neural network, in general, receives information in the form of external stimulation signals and performs some computation on this information, which is represented by changes of its state. The operation of the network can be viewed as a flow of excitement among the cells and the spatial distribution of this excitement represents the response of the network to the information received. This kind of operation is particularly relevant in the processing of image fields. On the other hand, in queueing networks, composed of a number of service station nodes, customers arrive from the outside world and spend some time in the network, during which they more from node to node, waiting and receiving service at each node visited. Following the external arrival pattern, the interconnection of nodes and the other network parameters, the operation of the network is characterized by a distribution of activity among the nodes.

Let us now consider a queueing network, where nodes represent cells and customers represent stimulations moving from cell to cell following the topology of the network. Our aim is to define the network's characteristics in a way to match those of the neural net model as much as possible. Our queueing network model is completely specified by the following assumptions:

- The network is composed of $M=N^2$ nodes arranged on an NxN rectangular grid, as in the previous case. Interconnections are expressed by means of a matrix R of routing probabilities: $r_{ij}$ ($1 \leq i, j \leq M$) represents the probability that a stimulation (customer) leaving node i will next visit node j. Since it is an open network, after visiting an arbitrary number of cells, stimulations may eventually leave the network. Let $r_{i0}$ denote the probability of leaving the network upon leaving node i. In what follows, we will assume that $r_{i0}=s$ for all nodes. In what concerns the routing probabilities $r_{ij}$, they are determined by the two interconnection schemata considered in the previous section (propagative and looping connections): each node i has two output nodes j, for which the routing probabilities are equally distributed. Thus, $r_{ij}=(1-s)/2$ for the two output nodes of i and equal to zero for all other nodes in the network.

- External stimulation arrivals follow a Poisson process with parameter $\lambda$ and are routed to the nodes according to the probability distribution $q_i$ ($1 \leq i \leq M$) as in the previous section.

- Stimulations receive a "service time" at each node visited. Service times are independent identically distributed random variables, which are exponentially distributed with parameter $y$. The time spent by a stimulation at a node depends also upon the "service discipline" adopted. We shall consider two types of service disciplines according to the general queueing network model [4]: the first-come-first-served (FCFS) discipline, where customers are served in the order of their arrival to the node, and the infinite-server (IS) discipline, where a customer's service is immediately assumed upon arrival to the node, as if there were always a server available for each arriving customer (the second type includes no waiting delay). We will refer to the above two types of nodes as type 1 and type 2 respectively. In either case, all nodes of the network will be of the same type.

The steady-state solution of the above network is a straightforward application of the general BCMP theorem [4] according to the

'simple assumptions considered. The state of the system is described by the vector $(k_1, k_2, \ldots, k_M)$, where $k_i$ is the number of customers present at node i. We first define the traffic intensity $\rho_i$ for each node i as

$$\rho_i = \lambda e_i / \gamma \quad , \quad i = 1, 2, \ldots, M \tag{1}$$

where the quantities $\{e_i\}$ are the solution of the following set of linear equations:

$$e_i = q_i + \sum_{j=1}^{M} e_j r_{ji} \quad , \quad i = 1, 2, \ldots, M \tag{2}$$

It can be easily seen that, in fact, $e_i$ represents the average number of visits a customer makes to node i during his sojourn in the network. The existence of a steady-state distribution for the system depends on the solution of the above set. Following the general theorem [4], the joint steady-state distribution takes the form of a product of independent distributions for the nodes:

$$p(k_1, k_2, \ldots, k_M) = p_1(k_1) p_2(k_2) \ldots p_M(k_M) \tag{3}$$

where

$$p_i(k_i) = \begin{cases} (1-\rho_i)\rho_i^{k_i} & \text{(Type 1)} \\ e^{-\rho_i} \dfrac{\rho_i^{k_i}}{k_i!} & \text{(Type 2)} \end{cases} \tag{4}$$

provided that the stability condition $\rho_i < 1$ is satisfied for type 1 nodes.

The product form solution of this type of network expresses the idea of global and local balance which is characteristic of ergodic Markov processes. We can then proceed to deriving the desired measure for each node in the network; we are interested in the probability of being active for each node, which can be interpreted as the probability that at least one customer is present at the node:

$$P(k_i > 0) = 1 - p_i(0) = \begin{cases} \rho_i & \text{(Type 1)} \\ 1 - e^{-\rho_i} & \text{(Type 2)} \end{cases} \tag{5}$$

The variation in space of the above quantity will be studied with respect to the corresponding measure obtained from simulation experiments for the neural network model.

## NUMERICAL AND SIMULATION EXAMPLES

Simulations and numerical solutions of the queueing network model were run for different values of the parameters. The network sizes considered are relatively small but can provide useful information on the spatial organization of the networks. For both types of service discipline discussed in the previous section, the approach followed yields a very good approximation of the network's organization in most regions of the rectangular grid. The choice of the probability s of leaving the network plays a critical role in the beha-

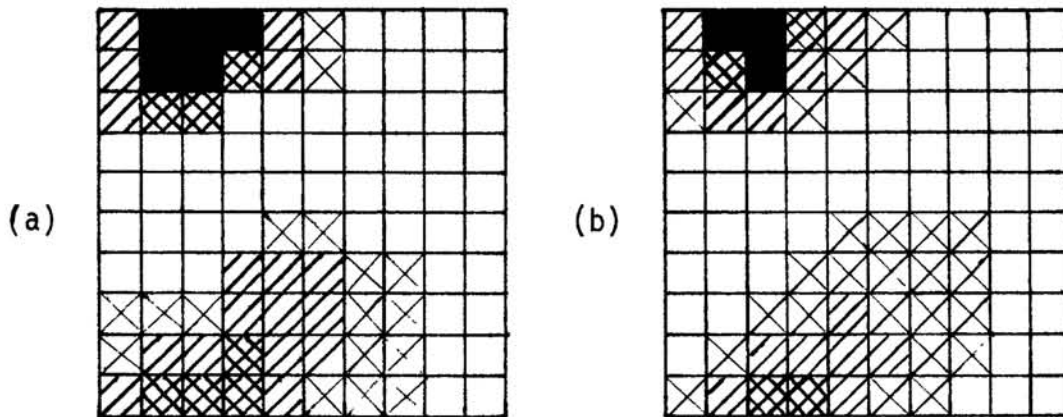

Fig.3. A 10x10 network with λ=1, γ=1 and propagative connections. External stimulations are uniformly distributed over a 3x3 square on the upper left corner of the grid. (a) simulation (b) Queueing network approach with s=0.05 and type 2 nodes.

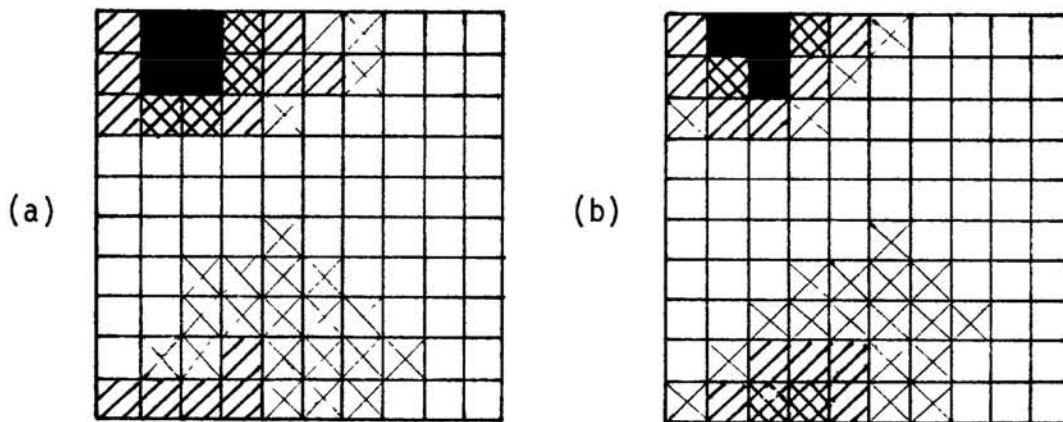

Fig.4. The network of Fig.3 with λ=2 (a) Simulation (b) Queueing network approach with s=0.08 and type 2 nodes.

viour of the queueing model, and must have a non-zero value in order for the network to be stable. Good results are obtained for very small values of s; in fact, this parameter represents the phenomenon of excitation being "lost" somewhere in the network. Graphical representations for various cases are shown in Figures 3-7. We have used a coloring of five "grey levels", defined by dividing into five segments the interval between the smallest and the largest value of the probability on the grid; the normalization is performed with respect to simulation results. This type of representation is less accurate than directly providing numerical values, but is more clear for describing the organization of the system. In each case, the results shown for the queueing model concern only one type of nodes, the one that best fits the simulation results, which is type 2 in the majority of cases examined. The graphical representation illustrates the structuring of the distribution of excitation on the grid in terms of functionally connected regions of high and low

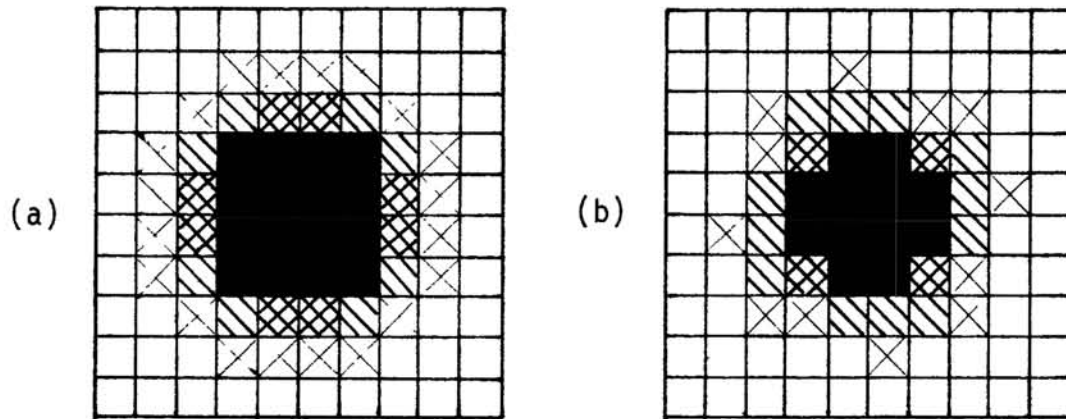

Fig.5. A 10x10 network with λ=1, γ=1 and looping connections. External stimulations are uniformly distributed over a 4x4 square on the center of the grid. (a) Simulation (b) Queueing network approach with s=0.07 and type 2 nodes.

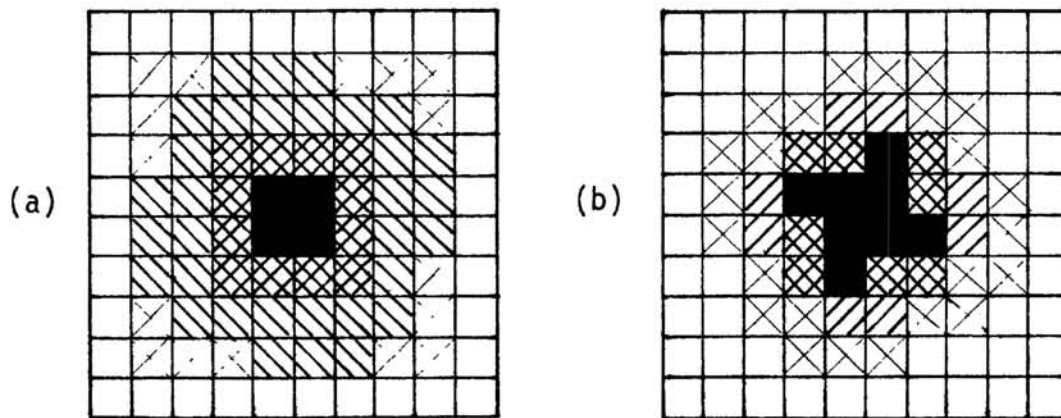

Fig.6. The network of Fig.5 with λ=0.5 (a) Simulation (b) Queueing network approach with s=0.03 and type 2 nodes.

excitation. We notice that clustering of nodes mainly follows the spatial distribution of external stimulations and is more sharply structured in the case of looping connections.

## CONCLUSION

We have developed in this paper a simple continuous-time probabilistic model of neural nets in an attempt to investigate their spatial organization. The model incorporates some of the main features of the McCulloch-Pitts "formal neurons" model and assumes boolean operation of the elementary cells. The steady-state behaviour of the model was approximated by means of a queueing network model with suitably chosen parameters. Results obtained from the solution of the above approximation were compared with simulation results of the initial model, which validate the approximation. This simplified approach is a first step in an attempt to study the organiza-

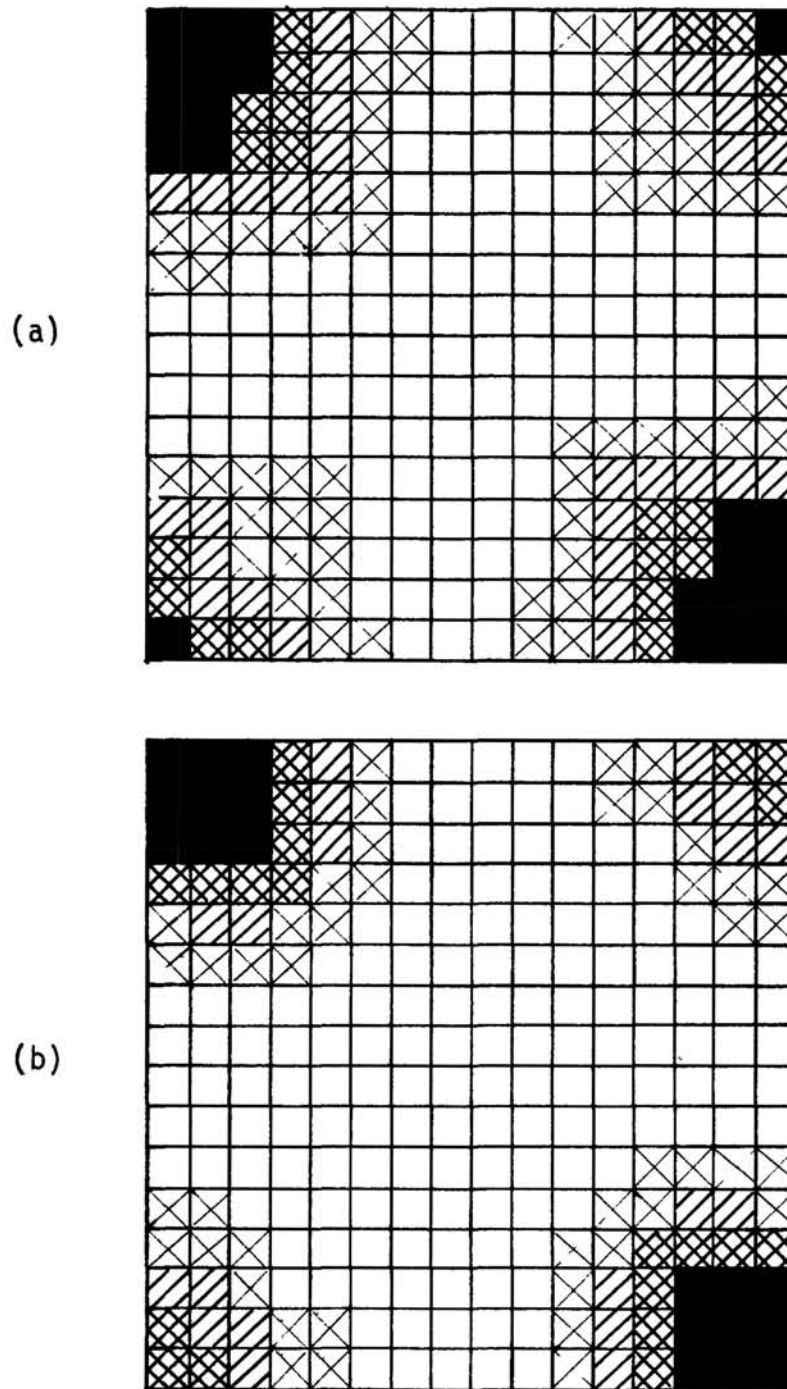

(a)

(b)

Fig.7. A 16x16 network with $\lambda=1$, $\gamma=1$ and looping connections. External stimulations are uniformly distributed over two 4x4 squares on the upper left and lower right corners of the grid. (a) Simulation (b) Queueing network approach with s=0.05 and type 1 nodes.

tional properties of neural nets by means of markovian modeling techniques.

## REFERENCES

1. W. S. McCulloch, W. Pitts, "A Logical Calculus of the Ideas Immanent in Nervous Activity", Bull. of Math. Biophysics 5, 115-133 (1943).
2. M. L. Minsky, Computation: Finite and Infinite Machines (Prentice Hall, 1967).
3. S. Kauffman, "Behaviour of Randomly Constructed Genetic Nets", in Towards a Theoretical Biology, Ed. C. H. Waddington (Edinburgh University Press, 1970).
4. F. Baskett, K. M. Chandy, R. R. Muntz, F. G. Palacios, "Open, Closed and Mixed Networks of Queues with Different Classes of Customers", J. ACM, 22 (1975).
5. H. Atlan, F. Fogelman-Soulié, J. Salomon, G. Weisbuch, "Random Boolean Networks", Cyb. and Syst. 12 (1981).
6. F. Folgeman-Soulié, E. Goles-Chacc, G. Weisbuch, "Specific Roles of the Different Boolean Mappings in Random Networks", Bull. of Math. Biology, Vol.44, No 5 (1982).
7. G. R. Cross, A. K. Jain, "Markov Random Field Texture Models", IEEE Trans. on PAMI, Vol. PAMI-5, No 1 (1983).
8. E. R. Kandel, J. H. Schwartz, Principles of Neural Science, (Elsevier, N.Y., 1985).
9. J. J. Hopfield, D. W. Tank, "Computing with Neural Circuits: A Model", Science, Vol. 233, 625-633 (1986).
10. Y. S. Abu-Mostafa, D. Psaltis, "Optical Neural Computers", Scient. Amer., 256, 88-95 (1987).
11. R. P. Lippmann, "An Introduction to Computing with Neural Nets", IEEE ASSP Mag. (Apr. 1987).
12. C. A. Mead, "Neural Hardware for Vision", Eng. and Scie. (June 1987).
13. E. Gelenbe, A. Stafylopatis, "Temporal Behaviour of Neural Networks", IEEE First Intern. Conf. on Neural Networks, San Diego, CA (June 1987).
14. L. Onural, "A Systematic Procedure to Generate Connected Binary Fractal Patterns with Resolution-varying Texture", Sec. Intern. Sympt. on Comp. and Inform. Sciences, Istanbul, Turkey (Oct. 1987).
